# An Approximate Inference Approach to Temporal Optimization in Optimal Control

**Konrad C. Rawlik**
School of Informatics
University of Edinburgh
Edinburgh, UK

**Marc Toussaint**
TU Berlin
Berlin, Germany

**Sethu Vijayakumar**
School of Informatics
University of Edinburgh
Edinburgh, UK

## Abstract

Algorithms based on iterative local approximations present a practical approach to optimal control in robotic systems. However, they generally require the temporal parameters (for e.g. the movement duration or the time point of reaching an intermediate goal) to be specified *a priori*. Here, we present a methodology that is capable of jointly optimizing the temporal parameters in addition to the control command profiles. The presented approach is based on a Bayesian canonical time formulation of the optimal control problem, with the temporal mapping from canonical to real time parametrised by an additional control variable. An approximate EM algorithm is derived that efficiently optimizes both the movement duration and control commands offering, for the first time, a practical approach to tackling generic via point problems in a systematic way under the optimal control framework. The proposed approach, which is applicable to plants with non-linear dynamics as well as arbitrary state dependent and quadratic control costs, is evaluated on realistic simulations of a redundant robotic plant.

## 1 Introduction

Control of sensorimotor systems, artificial or biological, is inherently both a spatial and temporal process. Not only do we have to specify where the plant has to move to but also when it reaches that position. In some control schemes, the temporal component is implicit; for example, with a PID controller, movement duration results from the application of the feedback loop, while in other cases it is explicit, like for example in finite or receding horizon optimal control approaches where the time horizon is set explicitly as a parameter of the problem [8, 13].

Although control based on an optimality criterion is certainly attractive, practical approaches for stochastic systems are currently limited to the finite horizon [9, 16] or first exit time formulation [14, 17]. The former does not optimize temporal aspects of the movement, i.e., duration or the time when costs for specific sub goals of the problem are incurred, assuming them as given *a priori*. However, how should one choose these temporal parameters? This question is non trivial and important even while considering a simple reaching problem. The solution generally employed in practice is to use a apriori fixed duration, chosen experimentally. This can result in not reaching the goal, having to use unrealistic range of control commands or excessive (wasteful) durations for short distance tasks. The alternative first exit time formulation, on the other hand, either assumes specific exit states in the cost function and computes the shortest duration trajectory which fulfils the task or assumes a time stationary task cost function and computes the control which minimizes the joint cost of movement duration and task cost [17, 1, 14]. This formalism is thus only directly applicable to tasks which do not require sequential achievement of multiple goals. Although this limitation could be overcome by chaining together individual time optimal single goal controllers, such a sequential approach has several drawbacks. First, if we are interested in placing a cost on overall movement duration, we are restricted to linear costs if we wish to remain time optimal. A second more important flaw is that

future goals should influence our control even before we have achieved the previous goal, a problem which we highlight during our comparative simulation studies.

A wide variety of successful approaches to address stochastic optimal control problems have been described in the literature [6, 2, 7]. The approach we present here builds on a class of approximate stochastic optimal control methods which have been successfully used in the domain of robotic manipulators and in particular, the iLQG [9] algorithm used by [10], and the *Approximate Inference Control* (AICO) algorithm [16]. These approaches, as alluded to earlier, are finite horizon formulations and consequently require the temporal structure of the problem to be fixed *a priori*. This requirement is a direct consequence of a fixed length discretization of the continuous problem and the structure of the temporally non-stationary cost function used, which binds incurrence of goal related costs to specific (discretised) time points. The fundamental idea proposed here is to reformulate the problem in canonical time and alternately optimize the temporal and spatial trajectories. We implement this general approach in the context of the approximate inference formulation of AICO, leading to an *Expectation Maximisation* (EM) algorithm where the E-Step reduces to the standard inference control problem. It is worth noting that due to the similarities between AICO, iLQG and other algorithms, e.g., DDP [6], the same principle and approach should be applicable more generally. The proposed approach provides an extension to the time scaling approach [12, 3] by considering the scaling for a complete controlled system, rather then a single trajectory. Additionally, it also extends previous applications of Expectation Maximisation algorithms for system identification of dynamical systems, e.g. [4, 5], which did not consider the temporal aspects.

## 2 Preliminaries

Let us consider a process with state $\mathbf{x} \in \mathbb{R}^{D_x}$ and controls $\mathbf{u} \in \mathbb{R}^{D_u}$ which is of the form

$$d\mathbf{x} = (\mathcal{F}(\mathbf{x}) + \mathbf{B}\mathbf{u})dt + d\xi \quad \langle d\xi d\xi^\top \rangle = \mathbf{Q} \tag{1}$$

with non-linear state dependent dynamics $\mathcal{F}$, control matrix $\mathbf{B}$ and Brownian motion $\xi$, and define a cost of the form

$$\mathcal{L}(\mathbf{x}(\cdot), \mathbf{u}(\cdot)) = \int_0^T \left[ \mathcal{C}(\mathbf{x}(t), t) + \mathbf{u}(t)^\top \mathbf{H}\mathbf{u}(t) \right] dt , \tag{2}$$

with arbitrary state dependent cost $\mathcal{C}$ and quadratic control cost. Note in particular that $T$, the trajectory length, is assumed to be known. The closed loop stochastic optimal control problem is to find the policy $\pi : \mathbf{x}(t) \rightarrow \mathbf{u}(t)$ given by

$$\pi^* = \underset{\pi}{\text{argmin}} \, \mathbb{E}_{\mathbf{x}, \mathbf{u} | \pi, \mathbf{x}(0)} \left\{ \mathcal{L}(\mathbf{x}(\cdot), \mathbf{u}(\cdot)) \right\} . \tag{3}$$

In practice, the continuous time problem is discretized into a fixed number of $K$ steps of length $\Delta_t$, leading to the discreet problem with dynamics

$$\mathsf{P}(\mathbf{x}_{k+1}|\mathbf{x}_k, \mathbf{u}_k) = \mathcal{N}(\mathbf{x}_{k+1}|\mathbf{x}_k + (\mathcal{F}(\mathbf{x}) + \mathbf{B}\mathbf{u})\Delta_t, \mathbf{Q}\Delta_t) , \tag{4}$$

where we use $\mathcal{N}(\cdot|a, A)$ to denote a Gaussian distribution with mean $a$ and covariance $A$, and cost

$$\mathcal{L}(\mathbf{x}_{1:K}, \mathbf{u}_{1:K}) = \mathcal{C}_K(\mathbf{x}_K) + \sum_{k=0}^{K-1} \left[ \Delta_t \mathcal{C}_k(\mathbf{x}_k) + \mathbf{u}_k^\top (\mathbf{H}\Delta_t)\mathbf{u}_k \right] . \tag{5}$$

Note that here we used the Euler Forward Method as the discretization scheme, which will prove advantageous if a linear cost on the movement duration is chosen, leading to closed form solution for certain optimization problems. However, in other cases, alternative discretisation methods could be used and indeed, be preferable.

### 2.1 Approximate Inference Control

Recently, it has been suggested to consider a Bayesian inference approach [16] to (discreet) optimal control problems formalised in Section 2. With the probabilistic trajectory model in (4) as a prior, an auxiliary (binary) dynamic random task variable $r_k$, with the associated likelihood

$$\mathsf{P}(r_k = 1|\mathbf{x}_k, \mathbf{u}_k) = \exp \left\{ -(\Delta_t \mathcal{C}_k(x_k) + \mathbf{u}_k^\top (\mathbf{H}\Delta_t)\mathbf{u}_k) \right\} , \tag{6}$$

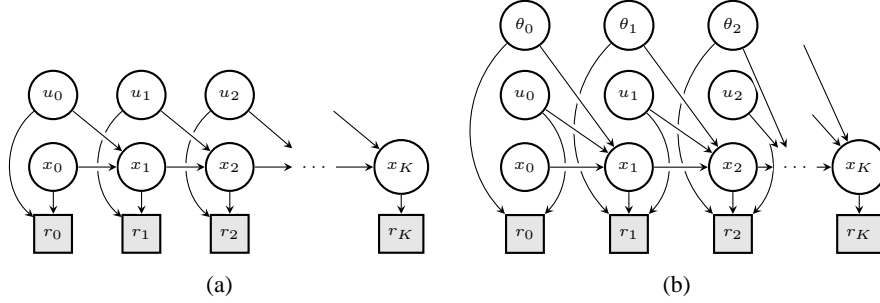

Figure 1: The graphical models for **(a)** standard inference control and **(b)** the AICO-T model with canonical time. Circle and square nodes indicate continous and discreet variables respectively. Shaded nodes are observed.

is introduced, i.e., we interpret the cost as a negative log likelihood of task fulfilment. Inference control consists of computing the posterior conditioned on the observation $r_{0:K} = 1$ within the resulting model (illustrated a graphical model in Fig. 1 (a)), and from it obtaining the *maximum a posteriori* (MAP) controls. For cases, where the process and cost are linear and quadratic in $\mathbf{u}$ respectively, the controls can be marginalised in closed form and one is left with the problem of computing the posterior

$$P(\mathbf{x}_{0:K}|r_{0:K} = 1) = \prod_k \mathcal{N}(\mathbf{x}_{k+1}|\mathbf{x}_k + \mathcal{F}(\mathbf{x}_k)\Delta_t, \mathbf{W}\Delta_t)\exp(-\Delta_t\mathcal{C}_k(x_k)) , \qquad (7)$$

with $\mathbf{W} := \mathbf{Q} + \mathbf{B}\mathbf{H}^{-1}\mathbf{B}^\top$.

As this posterior is in general not tractable, the AICO [16] algorithm computes a Gaussian approximation to the true posterior using an approximate message passing approach similar in nature to EP (details are given in supplementary material). The algorithm has been shown to have competitive performance when compared to iLQG [16].

## 3 Temporal Optimization for Optimal Control

Often the state dependent cost term $\mathcal{C}(\mathbf{x}, t)$ in (2) can be split into a set of costs which are incurred only at specific times: also referred to as goals, and others which are independent of time, that is

$$\mathcal{C}(\mathbf{x}, t) = \mathcal{J}(\mathbf{x}) + \sum_{n=1}^{N} \delta_{t=\hat{t}_n} \mathcal{V}_n(\mathbf{x}) . \qquad (8)$$

Classically, $\hat{t}_n$ refer to *real time* and are fixed. For instance, in a reaching movement, generally a cost that is a function of the distance to the target is incurred only at the final time $T$ while collision costs are independent of time and incurred throughout the movement. In order to allow the time point at which the goals are achieved to be influenced by the optimization, we will re-frame the goal driven part of the problem in a *canonical time* and in addition to optimizing the controls, also optimize the mapping from canonical to real time.

Specifically, we introduce into the problem defined by (1) & (2) the canonical time variable $\tau$ with the associated mapping

$$\tau = \beta(t) = \int_0^t \frac{1}{\theta(s)} ds , \quad \theta(\cdot) > 0 , \qquad (9)$$

with $\theta$ as an additional control. We also reformulate the cost in terms of the time $\tau$ as[1]

$$\mathcal{L}(\mathbf{x}(\cdot), \mathbf{u}(\cdot), \theta(\cdot)) = \sum_{n=1}^{N} \mathcal{V}_n(\mathbf{x}(\beta^{-1}(\hat{\tau}_n))) + \int_0^{\hat{\tau}_N} \mathcal{T}(\theta(s)) ds$$

$$+ \int_0^{\beta^{-1}(\hat{\tau}_N)} \left[\mathcal{J}(\mathbf{x}(t)) + \mathbf{u}(t)^\top \mathbf{H}\mathbf{u}(t)\right] dt , \qquad (10)$$

with $\mathcal{T}$ an additional cost term over the controls $\theta$ and the $\hat{\tau}_{1:N} \in \mathbb{R}$ assumed as given. Based on the last assumption, we are still required to choose the time point at which individual goals are achieved and how long the movement lasts; however, this is now done in terms of the canonical time and since by controlling $\theta$, we can change the real time point at which the cost is incurred, the exact choices for $\hat{\tau}_{1:N}$ are relatively unimportant. The real time behaviour is mainly specified by the additional cost term $\mathcal{T}$ over the new controls $\theta$ which we have introduced. Note that in the special case where $\mathcal{T}$ is linear, we have $\int_0^{\hat{\tau}_N} \mathcal{T}(\theta_s) ds = \mathcal{T}(T)$, i.e., $\mathcal{T}$ is equivalent to a cost on the total movement duration. Although here we will stick to the linear case, the proposed approach is also applicable to non-linear duration costs. We briefly note the similarity of the formulation to the canonical time formulation of [11] used in an imitation learning setting.

We now discretize the augmented system in canonical time with a fixed number of steps $K$. Making the arbitrary choice of a step length of 1 in $\tau$ induces, by (9), a sequence of steps in $t$ with length[2] $\Delta_k = \theta_k$. Using this time step sequence and (4) we can now obtain a discreet process in terms of the canonical time with an explicit dependence on $\theta_{0:K-1}$. Discretization of the cost in (10) gives

$$\mathcal{L}(\mathbf{x}_{1:K}, \mathbf{u}_{1:K}, \theta_{0:K-1}) = \sum_{n=1}^{N} \mathcal{V}_n(\mathbf{x}_{\hat{k}_n}) + \sum_{k=0}^{K-1} \left[ \mathcal{T}(\theta_k) + \mathcal{J}(\mathbf{x}_k)\theta_k + \mathbf{u}_k^\top \mathbf{H}\theta_k \mathbf{u}_k \right] , \qquad (11)$$

for some given $\hat{k}_{1:N}$. We now have a new formulation of the optimal control problem that no longer of the form of equations (4) & (5), e.g. (11) is no longer quadratic in the controls as $\theta$ is a control.

Proceeding as for standard inference control and treating the cost (11) as a neg-log likelihood of an auxiliary binary dynamic random variable, we obtain the inference problem illustrated by the Bayesian network in Figure 1(b). With controls $\mathbf{u}$ marginalised, our aim is now to find the posterior $P(\mathbf{x}_{0:K}, \theta_{0:K-1} | r_{0:K} = 1)$. Unfortunately, this problem is intractable even for the simplest case, e.g. LQG with linear duration cost. However, observing that for given $\theta_k$'s, the problem reduces to the standard case of Section 2.1 suggest restricting ourselves to finding the MAP estimate for $\theta_{0:K-1}$ and the associated posterior $P(\mathbf{x}_{0:K} | \theta_{0:K-1}^{\text{MAP}}, r_{0:K} = 1)$ using an EM algorithm. The solution is obtained by iterating the E- & M-Steps (see below) until the $\theta$'s have converged; we call this algorithm *AICO-T* to reflect the temporal aspect of the optimization.

### 3.1 E-Step

In general, the aim of the E-Step is to calculate the posterior over the unobserved variables, i.e. the trajectories, given the current parameter values, i.e. the $\theta^i$'s.

$$q^i(\mathbf{x}_{0:K}) = P(\mathbf{x}_{0:K} | r_{0:K} = 1, \theta_{0:K-1}^i) . \qquad (12)$$

However, as will be shown below we actually only require the expectations $\langle \mathbf{x}_k \mathbf{x}_k^\top \rangle$ and $\langle \mathbf{x}_k \mathbf{x}_{k+1}^\top \rangle$ during the M-Step. As these are in general not tractable, we compute a Gaussian approximation to the posterior, following an approximate message passing approach with linear and quadratic approximations to the dynamics and cost respectively [16] (for details, refer to supplementary material).

### 3.2 M-Step

In the M-Step, we solve

$$\theta_{0:K-1}^{i+1} = \underset{\theta_{0:K-1}}{\operatorname{argmax}} \, \mathcal{Q}(\theta_{0:K-1} | \theta_{0:K-1}^i) , \qquad (13)$$

with

$$\mathcal{Q}(\theta_{0:K-1} | \theta_{0:K-1}^i) = \langle \log P(\mathbf{x}_{0:K}, r_{0:K} = 1 | \theta_{0:K-1}) \rangle$$
$$= \sum_{k=0}^{K-1} \langle \log P(\mathbf{x}_{k+1} | \mathbf{x}_k, \theta_k) \rangle - \sum_{k=1}^{K-1} \left[ \mathcal{T}(\theta_k) + \theta_k \langle \mathcal{J}(\mathbf{x}_k) \rangle \right] + constant ,$$
$$(14)$$

where $\langle \cdot \rangle$ denotes the expectation with respect to the distribution calculated in the E-Step, i.e., the posterior $q^i(\mathbf{x}_{0:K})$ over trajectories given the previous parameter values. The required expectations,

$\langle \mathcal{J}(\mathbf{x}_k) \rangle$ and

$$\langle \log \mathsf{P}(\mathbf{x}_{k+1}|\mathbf{x}_k, \theta_k) \rangle = -\frac{D_x}{2} \log |\widetilde{\mathbf{W}}_k| - \frac{1}{2} \left\langle (\mathbf{x}_{k+1} - \widetilde{\mathcal{F}}(\mathbf{x}_k))^\top \widetilde{\mathbf{W}}_k^{-1} (\mathbf{x}_{k+1} - \widetilde{\mathcal{F}}(\mathbf{x}_k)) \right\rangle , \quad (15)$$

with $\widetilde{\mathcal{F}}(\mathbf{x}_k) = \mathbf{x}_k + \mathcal{F}(\mathbf{x}_k)\theta_k$ and $\widetilde{\mathbf{W}}_k = \theta_k \mathbf{W}$, are in general not tractable. Therefore, we take approximations

$$\mathcal{F}(\mathbf{x}_k) \approx \mathbf{a}_k + \mathbf{A}_k \mathbf{x}_k \quad \text{and} \quad \mathcal{J}(\mathbf{x}_k) \approx \frac{1}{2} \mathbf{x}_k^\top \mathbf{J}_k \mathbf{x}_k - \mathbf{j}_k^\top \mathbf{x}_k , \quad (16)$$

choosing the mean of $q^i(\mathbf{x}_k)$ as the point of approximation, consistent with the equivalent approximations made in the E-Step. Under these approximations, it can be shown that, up to additive terms independent of $\theta$,

$$\mathcal{Q}(\theta_{0:K-1}|\theta_{0:K-1}^i) = - \sum_{k=0}^{K-1} \left[ \frac{D_x}{2} \log |\widetilde{\mathbf{W}}_k| + \mathcal{T}(\theta_k) + \frac{1}{2} \mathrm{Tr}(\widetilde{\mathbf{W}}_k^{-1} \langle \mathbf{x}_{k+1} \mathbf{x}_{k+1}' \rangle) \right.$$
$$- \mathrm{Tr}(\widetilde{\mathbf{A}}_k' \widetilde{\mathbf{W}}_k^{-1} \langle \mathbf{x}_{k+1} \mathbf{x}_k' \rangle) + \frac{1}{2} \mathrm{Tr}(\widetilde{\mathbf{A}}_k \widetilde{\mathbf{W}}_k^{-1} \widetilde{\mathbf{A}}_k' \langle \mathbf{x}_k \mathbf{x}_k' \rangle) + \tilde{\mathbf{a}}_k^\top \widetilde{\mathbf{W}}_k^{-1} \widetilde{\mathbf{A}}_k \langle \mathbf{x}_k \rangle$$
$$\left. + \frac{1}{2} \tilde{\mathbf{a}}_k^\top \widetilde{\mathbf{W}}_k^{-1} \tilde{\mathbf{a}}_k + \theta_k \left[ \frac{1}{2} \mathrm{Tr}(\mathbf{J}_k \langle \mathbf{x}_k \mathbf{x}_k^\top \rangle) - \mathbf{j}_k \langle \mathbf{x}_k \rangle \right] \right] ,$$

with $\tilde{\mathbf{a}}_k^\top = \theta_k \mathbf{a}_k$, $\widetilde{\mathbf{A}}_k = \mathbf{I} + \theta_k \mathbf{A}_k$ and taking partial derivatives leads to

$$\frac{\partial \mathcal{Q}}{\partial \theta_k} = \frac{1}{2} \theta_k^{-2} \mathrm{Tr} \left( \mathbf{W}^{-1} (\langle \mathbf{x}_{k+1} \mathbf{x}_{k+1}^\top \rangle - 2 \langle \mathbf{x}_{k+1} \mathbf{x}_k^\top \rangle + \langle \mathbf{x}_k \mathbf{x}_k^\top \rangle) \right) - \frac{D_x^2}{2} \theta_k^{-1}$$
$$- \frac{1}{2} \left[ \mathrm{Tr}(\mathbf{A} \mathbf{W}^{-1} \mathbf{A}^\top \langle \mathbf{x}_k \mathbf{x}_k^\top \rangle) + 2 \left. \frac{d\mathcal{T}}{d\theta} \right|_{\theta_k} + \mathbf{a}_k^\top \mathbf{W}^{-1} \mathbf{a}_k + 2 \mathbf{a}_k^\top \mathbf{W}^{-1} \mathbf{A}_k \langle \mathbf{x}_k \rangle \right. \quad (17)$$
$$\left. + \mathrm{Tr}(\mathbf{J}_k \langle \mathbf{x}_k \mathbf{x}_k^\top \rangle) - 2 \mathbf{j}_k \langle \mathbf{x}_k \rangle \right] .$$

In the general case, we can now use gradient ascent to improve the $\theta$'s. However, in the specific case where $\mathcal{T}$ is a linear function of $\theta$, we note that $0 = \frac{\partial \mathcal{Q}}{\partial \theta_k}$ is a quadratic in $\theta_k^{-1}$ and the unique extremum under the constraint $\theta_k > 0$ can be found analytically.

### 3.3 Practical Remarks

The performance of the algorithm can be greatly enhanced by using the result of the previous E-Step as initialisation for the next one. As this is likely to be near the optimum with the new temporal trajectory, AICO converges within only a few iterations. Additionally, in practise it is often sufficient to restrict the $\theta_k$'s between goals to be constant, which is easily achieved as $\mathcal{Q}$ is a sum over the $\theta$'s.

The proposed algorithms leads to a variation of discretization step length which can be a problem. For one, the approximation error increases with the step length which may lead to wrong results. On the other hand, the algorithm may lead to control frequencies which are not achievable in practice. In general, a fixed control signal frequency may be prescribed by the hardware system. In practice, $\theta$'s can be kept in a prescribed range by adjusting the number of discretization steps $K$ after an M-Step.

Finally, although we have chosen to express the time cost in terms of a function of the $\theta$'s, often it may be desirable to consider a cost directly over the duration $T$. Noting that $T = \sum \theta_k$, all that is required is to replace $\frac{d\mathcal{T}}{d\theta}$ with $\frac{\partial \mathcal{T}(\sum \theta)}{\partial \theta_k}$ in (17).

## 4 Experiments

The proposed algorithm was evaluated in simulation. As a basic plant, we used a kinematic simulation of a 2 degrees of freedom (DOF) planar arm, consisting of two links of equal length. The state of the plant is given by $\mathbf{x} = (\mathbf{q}, \dot{\mathbf{q}})$, with $\mathbf{q} \in \mathbb{R}^2$ the joint angles and $\dot{\mathbf{q}} \in \mathbb{R}^2$ associated angular

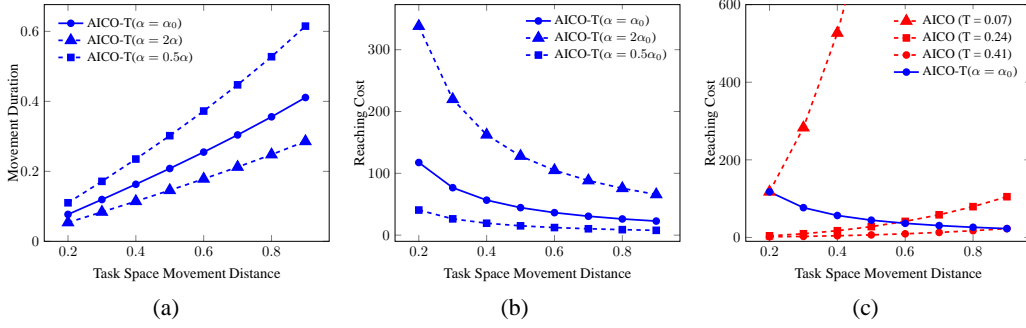

Figure 2: Temporal scaling behaviour using AICO-T. **(a & b)** Effect of changing time-cost weight $\alpha$, (effectively the ratio between reaching cost and duration cost) on **(a)** duration and **(b)** reaching cost (control + state cost). **(c)** Comparison of reaching costs (control + error cost) for AICO-T and a fixed duration approach, i.e. AICO.

velocities. The controls $\mathbf{u} \in \mathbb{R}^2$ are the joint space accelerations. We also added some iid noise with small diagonal covariance.

For all experiments, we used a quadratic control cost and the state dependent cost term:

$$\mathcal{V}(\mathbf{x}_k) = \sum_i \delta_{k=\hat{k}_i} (\phi_i(\mathbf{x}_k) - \mathbf{y}_i^*)^\top \Lambda_i (\phi_i(\mathbf{x}_k) - \mathbf{y}_i^*) , \tag{18}$$

for some given $\hat{k}_i$ and employed a diagonal weight matrix $\Lambda_i$ while $\mathbf{y}_i^*$ represented the desired state in task space. For point targets, the task space mapping is $\phi(\mathbf{x}) = (x, y, \dot{x}, \dot{y})^\top$, i.e., the map from $\mathbf{x}$ to the vector of end point positions and velocities in task space coordinates. The time cost was linear, that is, $\mathcal{T}(\theta) = \alpha\theta$.

## 4.1  Variable Distance Reaching Task

In order to evaluate the behaviour of AICO-T we applied it to a reaching task with varying start-target distance. Specifically, for a fixed start point we considered a series of targets lying equally spaced along a line in task space. It should be noted that although the targets are equally spaced in task space and results are shown with respect to movement distance in task space, the distances in joint space scale non linearly. The state cost (18) contained a single term incurred at the final discrete step with $\Lambda = 10^6 \cdot \mathbf{I}$ and the control cost were given by $\mathbf{H} = 10^4 \cdot \mathbf{I}$. Fig. 2(a & b) shows the movement duration $(= \sum \theta_k)$ and standard reaching cost[3] for different temporal-cost parameters $\alpha$ (we used $\alpha_0 = 2 \cdot 10^7$), demonstrating that AICO-T successfully trades-off the movement duration and standard reaching cost for varying movement distances. In Fig. 2(c), we compare the reaching costs of AICO-T with those obtained with a fixed duration approach, in this case AICO. Note that although with a fixed, long duration (e.g., AICO with duration T=0.41) the control and error costs are reduced for short movements, these movements necessarily have up to $4\times$ longer durations than those obtained with AICO-T. For example for a movement distance of 0.2 application of AICO-T results in a optimised movement duration of 0.07 (cf. Fig. 2(a)), making the fixed time approach impractical when temporal costs are considered. Choosing a short duration on the other hand (AICO (T=0.07)) leads to significantly worse costs for long movements. We further emphasis that the fixed durations used in this comparison were chosen post hoc by exploiting the durations suggested by AICO-T  in absence of this, there would have been no practical way of choosing them apart from experimentation. Furthermore, we would like to highlight that, although the results suggests a simple scaling of duration with movement distance, in cluttered environments and plants with more complex forward kinematics, an efficient decision on the movement duration cannot be based only on task space distance.

## 4.2  Via Point Reaching Tasks

We also evaluated the proposed algorithm in a more complex via point task. The task requires the end-effector to reach to a target, having passed at some point through a given second target, the

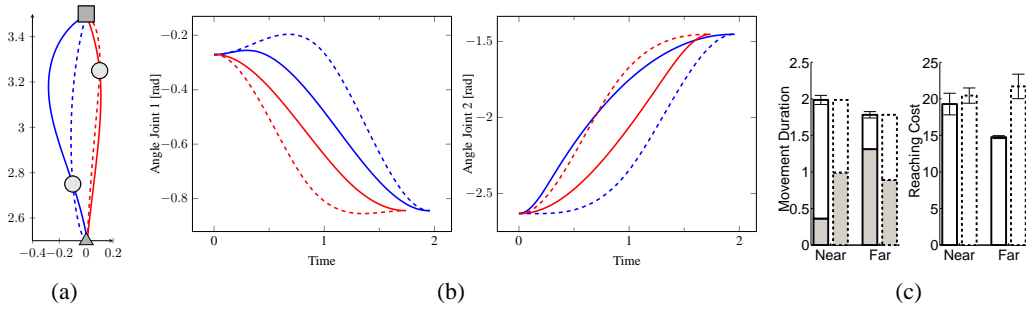

<div align="center">(a)          (b)          (c)</div>

Figure 3: Comparison of AICO-T (solid) to the common modelling approach, using AICO, (dashed) with fixed times on a via point task. **(a)** End point task space trajectories for two different via points (circles) obtained for a fixed start point (triangle). **(b)** The corresponding joint space trajectories. **(c)** Movement durations and reaching costs (control + error costs) from 10 random start points. The proportion of the movement duration spend before the via point is shown in light gray (mean in the AICO-T case).

via point. This task is of interest as it can be seen as an abstraction of a diverse range of complex sequential tasks that requires one to achieve a series of sub-tasks in order to reach a final goal. This task has also seen some interest in the literature on modeling of human movement using the optimal control framework, e.g., [15]. Here the common approach is to choose the time point at which one passes the via point such as to divide the movement duration in the same ratio as the distances between the start point, via point and end target. This requires on the one hand prior knowledge of these movement distances and on the other, makes the implicit assumption that the two movements are in some sense independent.

In a first experiment, we demonstrate the ability of our approach to solve such sequential problems, adjusting movement durations between sub goals in a principled manner, and show that it improves upon the standard modelling approach. Specifically, we apply AICO-T to the two via point problems illustrated in Fig. 3(a) with randomised start states[4]. For comparison, we follow the standard modeling approach and apply AICO to compute the controller. We again choose the movement duration for the standard case post hoc to coincide with the mean movement duration obtained with AICO-T for each of the individual via point tasks. Each task is expressed using a cost function consisting of two point target cost terms. Specifically, (18) takes the form

$$\mathcal{V}(\mathbf{x}_k) = \delta_{k=\frac{K}{2}}(\phi(\mathbf{x}_k) - \mathbf{y}_v^*)^\top \Lambda_v (\phi(\mathbf{x}_k) - \mathbf{y}_v^*) + \delta_{k=K}(\phi(\mathbf{x}_k) - \mathbf{y}_e^*)^\top \Lambda_e (\phi(\mathbf{x}_k) - \mathbf{y}_e^*), \quad (19)$$

with $K$ the number of discrete steps and diagonal matrices $\Lambda_v = \text{diag}(\lambda_{pos}, \lambda_{pos}, 0, 0)$, $\Lambda_e = \text{diag}(\lambda_{pos}, \lambda_{pos}, \lambda_{vel}, \lambda_{vel})$, where $\lambda_{pos} = 10^5$ & $\lambda_{vel} = 10^7$ and vectors $\mathbf{y}_v^* = (\cdot, \cdot, 0, 0)^\top$, $\mathbf{y}_e^* = (\cdot, \cdot, 0, 0)^\top$ desired states for individual via point and target, respectively. Note that the cost function does not penalise velocity at the via point but encourages the stopping at the target. While admittedly the choice of incurring the via point cost at the middle of the movement ($\frac{K}{2}$) is likely to be a sub-optimal choice for the standard approach, one has to consider that in more complex task spaces, the relative ratio of movement distances may not be easily accessible and one may have to resort to the most intuitive choice for the uninformed case as we have done here. Note that although for AICO-T this cost is incurred at the same discrete step, we allow $\theta$ before and after the via point to differ, but constrain them to be constant throughout each part of the movement, hence, allowing the cost to be incurred at an arbitrary point in real time. We sampled the initial position of each joint independently from a Gaussian distribution with a variance of $3°$. In Fig. 3(a&b), we show *maximum a posteriori* (MAP) trajectories in task space and joint space for controllers computed for the mean initial state. Interestingly, although the end point trajectory for the *near* via point produced by AICO-T may look sub-optimal than that produced by the standard AICO algorithm, closer examination of the joint space trajectories reveal that our approach results in more efficient actuation trajectories. In Fig. 3(c), we illustrate the resulting average movement durations and costs of the mean trajectories. As can be seen, AICO-T results in the expected passing times for the two via points, i.e. early vs. late in the movement for the near and far via point, respectively. This directly leads to a lower incurred cost compared to un-optimised movement durations.

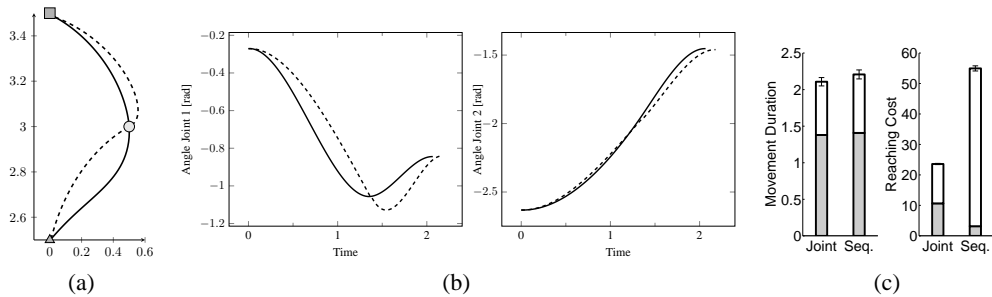

Figure 4: Joint (solid) vs. sequential (dashed) optimisation using AICO-T for a sequential (via point) task. **(a)** Task space trajectories for a fixed start point (triangle). Viapoint and target are indicated by the circle and square, respectively. **(b)** The corresponding joint space trajectories. **(c)** The movement durations and reaching costs (control + error cost) for 10 random start points. The mean proportion of the movement duration spend before the via point is shown in light gray.

In order to highlight the shortcomings of sequential time optimal control, next we compare planning a complete movement over sequential goals to planning a sequence of individual movements. Specifically, using AICO-T, we compare planning the whole via point movement (*joint planner*) to planning a movement from the start to the via point followed by a second movement from the end point of the first movement (n.b. not from the via point) to the end target (*sequential planner*). The joint planner used the same cost function as the previous experiment. For the sequential planner, each of the two sub-trajectories had half the number of discrete time steps of the joint planner and the cost functions were given by appropriately splitting (19), i.e.,

$$\mathcal{V}^1(\mathbf{x}_k) = \delta_{k=\frac{K}{2}}(\phi(\mathbf{x}_k)-\mathbf{y}_v^*)^\top \Lambda_v(\phi(\mathbf{x}_k)-\mathbf{y}_v^*) \quad \text{and} \quad \mathcal{V}^2(\mathbf{x}_k) = \delta_{k=\frac{K}{2}}(\phi(\mathbf{x}_k)-\mathbf{y}_e^*)^\top \Lambda_e(\phi(\mathbf{x}_k)-\mathbf{y}_e^*),$$

with $\Lambda_v, \Lambda_e, \mathbf{y}_v^*, \mathbf{y}_e^*$ as for (19). The start states were sampled according to the distribution used in the last experiment and in Fig. 4(a&b), we plot the MAP trajectories for the mean start state, in task as well as joint space. The results illustrate that sequential planning leads to sub-optimal results as it does not take future goals into consideration. This leads directly to a higher cost (c.f. Fig. 4(c)), calculated from trials with randomised start state. One should however note that this effect would be less pronounced if the cost required stopping at the via point, as it is the velocity away from the end target which is the main problem for the sequential planner.

## 5  Conclusion

The contribution of this paper is a novel method for jointly optimizing a movement trajectory and its time evolution (temporal scale and duration) in the stochastic optimal control framework. As a special case, this solves the problem of an unknown goal horizon and the problem of trajectory optimization through via points when the timing of intermediate constraints is unknown and subject to optimization. Both cases are of high relevance in practical robotic applications where pre-specifying a goal horizon by hand is common practice but typically lacks justification.

The method was derived in the form of an Expectation-Maximization algorithm where the E-step addresses the stochastic optimal control problem reformulated as an inference problem and the M-step re-adapts the time evolution of the trajectory. In principle, the proposed framework can be applied to extend any algorithm that – directly or indirectly – provides us with an approximate trajectory posterior in each iteration. AICO [16] does so directly in terms of a Gaussian approximation; similarly, the local LQG solution implicit in iLQG [9] can, with little extra computational cost, be used to compute a Gaussian posterior over trajectories. For algorithms like DDP [6], which do not lead to an LQG approximation, we can employ the Laplace method to obtain Gaussian posteriors or adjust the M-Step for the non-Gaussian posterior. We demonstrated the algorithm on a standard reaching task with and without via points. In particular, in the via point case, it becomes obvious that fixed horizon methods and sequenced first exit time methods cannot find equally efficient motions as the proposed method.

## Footnotes

[1]Note that as $\beta$ is strictly monotonic and increasing, the inverse function $\beta^{-1}$ exists

[2]under the assumption of constant $\theta(\cdot)$ during each step

[3]n.b. the *standard reaching cost* is the sum of control costs and cost on the endpoint error, without taking duration into account, i.e., (11) without the $\mathcal{T}(\theta)$ term.

[4]For the sake of clarity, Fig. 3(a&b) show MAP trajectories of controllers computed for the mean start state.

# References

[1] David Barber and Tom Furmston. Solving deterministic policy (PO)MDPs using expectation-maximisation and antifreeze. In *European Conference on Machine Learning (LEMIR workshop)*, 2009.

[2] Marc Peter Deisenroth, Carl Edward Rasmussen, and Jan Peters. Gaussian process dynamic programming. *Neurocomputing*, 72(7-9):1508 – 1524, 2009.

[3] Yu-Yi Fu, Chia-Ju Wu, Kuo-Lan Su, and Chia-Nan Ko. A time-scaling method for near-time-optimal control of an omni-directional robot along specified paths. *Artificial Life and Robotics*, 13(1):350–354, 2008.

[4] Z Ghahramani and G Hinton. Parameter estimation for linear dynamical systems. Technical Report CRG-TR-96-2, University of Toronto, 1996.

[5] Z Ghahramani and S Roweis. Learning nonlinear dynamical systems using an em algorithm. In *Advances in Neural Information Processing Systems*, volume 11, Nov 1999.

[6] D Jacobson and D Mayne. *Differential Dynamic Programming*. Elsevier, 1970.

[7] Hilbert J. Kappen. A linear theory for control of non-linear stochastic systems. *Physical Review Letters*, 95(20):200201, 2005.

[8] Donald E. Kirk. *Optimal Control Theory - An Introduction*. Prentice-Hall, 1970.

[9] Weiwei Li and Emanuel Todorov. An iterative optimal control and estimation design for non-linear stochastic system. In *Proc. of the 45th IEEE Conference on Decision and Control*, 2006.

[10] Djordje Mitrovic, Sho Nagashima, Stefan Klanke, Takamitsu Matsubara, and Sethu Vijayakumar. Optimal feedback control for anthropomorphic manipulators. In *Proc. IEEE International Conference on Robotics and Automation (ICRA 2010)*, 2010.

[11] Peter Pastor, Heiko Hoffmann, Tamim Asfour, and Stefan Schaal. Learning and generalization of motor skills by learning from demonstration. In *Proc. IEEE International Conference on Robotics and Automation (ICRA 2010)*, Feb 2010.

[12] Gideon Sahar and John M. Hollerbach. Planning of minimum- time trajectories for robot arms. *The International Journal of Robotics Research*, 5(3):90–100, 1986.

[13] Robert F. Stengel. *Optimal Control and Estimation*. Dover Publications, 1986.

[14] Emanuel Todorov. Compositionality of optimal control laws. In *Advances in Neural Information Processing Systems*, volume 22, 2009.

[15] Emanuel Todorov and Michael Jordan. Optimal feedback control as a theory of motor coordination. *Nature Neuroscience*, 5(11):1226–1235, 2002.

[16] Marc Toussaint. Robot trajectory optimization using approximate inference. In *Proc. of the 26 th International Conference on Machine Learning (ICML 2009)*, 2009.

[17] Marc Toussaint and Amos Storkey. Probabilistic inference for solving discrete and continuous state Markov Decision Processes. In *Proc. of the 23nd Int. Conf. on Machine Learning (ICML 2006)*, pages 945–952, 2006.

